# Classifying Hand Gestures with a View-based Distributed Representation

**Trevor J. Darrell**
Perceptual Computing Group
MIT Media Lab

**Alex P. Pentland**
Perceptual Computing Group
MIT Media Lab

## Abstract

We present a method for learning, tracking, and recognizing human hand gestures recorded by a conventional CCD camera without any special gloves or other sensors. A view-based representation is used to model aspects of the hand relevant to the trained gestures, and is found using an unsupervised clustering technique. We use normalized correlation networks, with dynamic time warping in the temporal domain, as a distance function for unsupervised clustering. Views are computed separably for space and time dimensions; the distributed response of the combination of these units characterizes the input data with a low dimensional representation. A supervised classification stage uses labeled outputs of the spatio-temporal units as training data. Our system can correctly classify gestures in real time with a low-cost image processing accelerator.

## 1 INTRODUCTION

Gesture recognition is an important aspect of human interaction, either interpersonally or in the context of man-machine interfaces. In general, there are many facets to the "gesture recognition" problem. Gestures can be made by hands, faces, or one's entire body; they can be static or dynamic, person-specific or cross-cultural. Here we focus on a subset of the general task, and develop a method for interpreting dynamic hand gestures generated by a specific user. We pose the problem as one of spotting instances of a set of known (previously trained) gestures. In this context, a gesture can be thought of as a set of hand views observed over time, or simply as a sequence of images of hands over time. These images may occur at different temporal rates, and the hand may have different spatial

offset or gross illumination condition. We would like to achieve real- or near real-time performance with our system, so that it can be used interactively by users.

To achieve this level of performance, we take advantage of the principle of using only as much "representation" as needed to perform the task. Hands are complex, 3D articulated structures, whose kinematics and dynamics are difficult to fully model. Instead of performing explicit model-based reconstruction, and attempting to extract these 3D model parameters (for example see [4, 5, 6]), we use a simpler approach which uses a set of 2D views to represent the object. Using this approach we can perform recognition on objects which are either too difficult to model or for which a model recovery method is not feasible. As we shall see below, the view-based approach affords several advantages, such as the ability to form a sparse representation that only models the poses of the hands that are relevant to the desired recognition tasks, and the ability to learn the relevant model directly from the data using unsupervised clustering.

## 2  VIEW-BASED REPRESENTATION

Our task is to recognize spatio-temporal sequences of hand images. To reduce the dimensionality of the matching involved, we find a set of view images and a matching function such that the set of match scores of a new image with the view images is adequate for recognition. The matching function we use is the normalized correlation between the image and the set of learned spatial views.

Each view represents a different pose of the object being tracked or recognized. We construct a set of views that "spans" the set of images seen in the training sequences, in the sense that at least one view matches every frame in the sequence (given a distance metric and threshold value). We can then use the view with the maximum score (minimum distance) to localize the position of the object during gesture performance, and use the ensemble response of the view units (at the location of maximal response) to characterize the actual pose of the object. Each model is based on one or more example images of a view of an object, from which mean and variance statistics about each pixel in the view are computed.

The general idea of view-based representation has been advocated by Ullman [12] and Poggio [9] for representing 3-D objects by interpolating between a small set of 2-D views. Recognition using views was analyzed by Breuel, who established bounds on the number of views needed for a given error rate [3]. However the view-based models used in these approaches rely on a feature-based representation of an image, in which a "view" is the list of vertex locations of semantically relevant features. The automatic extraction of these features is not a fully solved problem. (See [2] for a nearly automated system of finding corresponding points and extracting views.)

Most similar to our work is that of Murase and Nayar[8] and Turk[11] which use low-order eigenvectors to reduce the dimensionality of the signal and perform recognition. Our work differs from theirs in that we use normalized-correlation model images instead of eigenfunctions and can thus localize the hand position more directly, and we extend into the temporal domain, recognizing image sequences of gestures rather than static poses.

A particular view model will have a range of parameter values of a given transformation (e.g., rotation, scale, articulation) for which the correlation score shows a roughly convex "tuning curve". If we have a set of view models which sample the transformation parameter

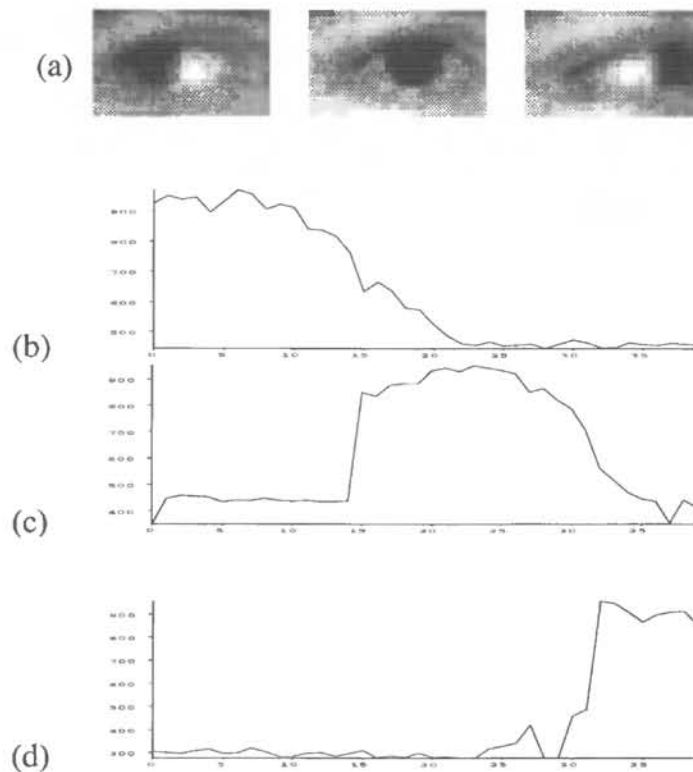

Figure 1: (a) Three views of an eyeball: +30, 0, and −30 of gaze angle. (a) Normalized correlation scores of the +30 degree view model when tracking a eyeball rotating from approximately −30 to +30 degrees of gaze angle. (b) Score for 0 degree view model. (c) Score for −30 degree model.

finely enough, it is possible to infer the actual transform parameters for new views by examining the *set* of model correlation scores. For example, Figure 1a shows three views of an eyeball that could be used for gaze tracking; one looking 30 degrees left, one looking center-on, and one looking 30 degrees to the right. The three views span a ±30 degree subspace of the gaze direction parameter. Figure 1(b,c,d) shows the normalized correlation score for each view model when tracking a rotating eyeball. Since the tuning curves produced by these models are fairly broad with respect to gaze angle, one could interpolate from their responses to obtain a good estimate of the true angle.

When objects are non-rigid, either constructed out of flexible materials or an articulated collection of rigid parts (like a hand), then the dimensionality of the space of possible views becomes much larger. Full coverage of the view space in these cases is usually not possible since enumerating it even with very coarse sampling would be prohibitively expensive in terms of storage and search computation required. However, many parts of a high dimensional view space may never be encountered when processing real sequences, due to unforeseen additional constraints. These may be physical (some joints may not be completely independent), or behavioral (some views may never be used in the actual communication between user and machine). A major advantage of our adaptive scheme is that it has no difficulty with sparse view spaces, and derives from the data which regions of the space are full.

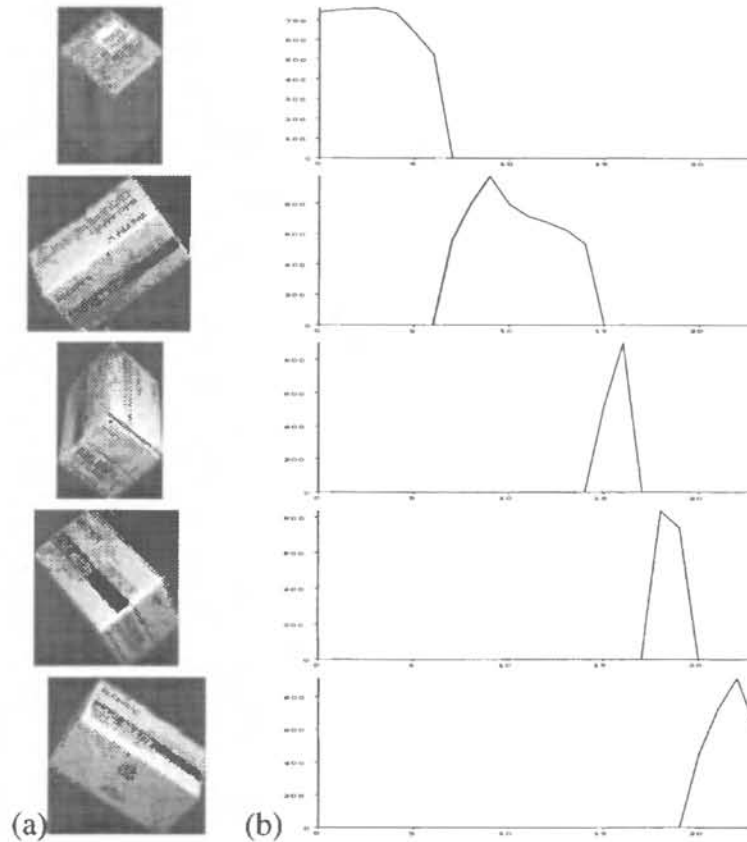

Figure 2: (a) Models automatically acquired from a sequence of images of a rotating box. (b) Normalized correlation scores for each model as a function of image sequence frame number.

## 3    UNSUPERVISED LEARNING OF VIEW UNITS

To derive a set of new view models, we use a simple form of unsupervised clustering in which the first example forms a new view, and subsequent examples that are below a distance threshold are merged into the nearest existing view. A new view is created when an example is below the threshold distance for all views in the current set, but is above a base threshold which establishes that the object is still (roughly) being tracked. Over time, this "follow-the-leader" algorithm results in a family of view models that sample the space of object poses in the training data. This method is similar to those commonly used in vector quantization [7]. Variance statistics are updated for each model pixel, and can be used to exclude unreliable points from the correlation computation.

For simple objects and transformations, this adaptive scheme can build a model which adequately covers the entire space of possible views. For example, for a convex rigid body undergoing a 1D rotation with fixed relative illumination, a relatively small number of view models can suffice to track and interpolate the position of the object at any rotation. Figures 2 illustrates this with a simple example of a rotating box. The adaptive tracking scheme was run with a camera viewing a box rotating about a fixed axis. Figure 2a shows the view models in use when the algorithm converged, and all possible rotations were matched with score greater than $\theta_1$. To demonstrate the tuning properties of each model under rotation, Figure 2b shows the correlation scores for each model plotted as a function of input frame

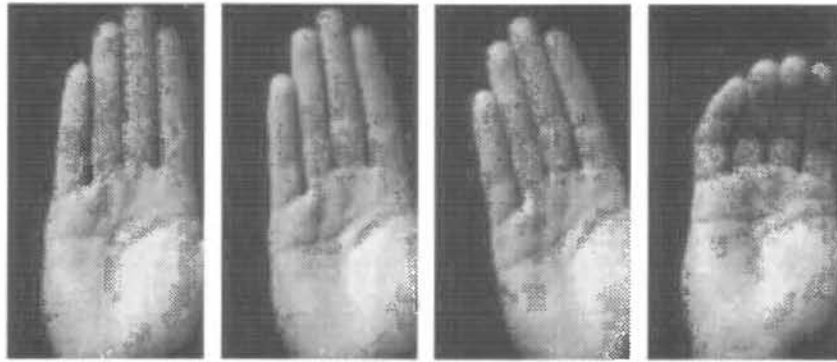

Figure 3: Four spatial views found by unsupervised clustering method on sequence containing two hand-waving gestures: side-to-side and up-down.

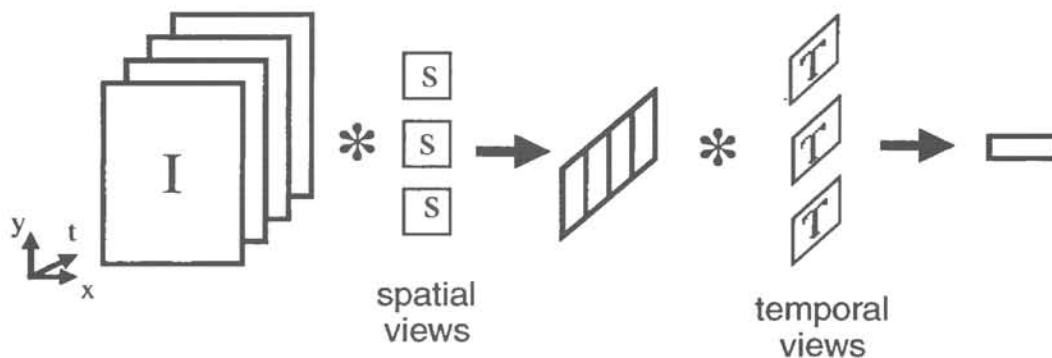

Figure 4: Overview of unsupervised clustering stage to learn spatial and temporal views. An input image sequence is reduced to sequence of feature vectors which record the maximum value in a normalized correlation network corresponding to each spatial view. A similar process using temporal views reduces the spatial feature vectors to a single spatio-temporal feature vector.

number of a demonstration sequence. In this sequence the box was held fixed at its initial position for the first 5 frames, and then rotated continuously from 0 to 340 degrees. The responses of each model are broadly tuned as a function of object angle, with a small number of models sufficing to represent/interpolate the object at all rotations (at least about a single axis).

We ran our spatial clustering method on images of hands performing two different "waving" gestures. One gesture was a side-to-side wave, with the fingers rigid, and the other was an up-down wave, with the wrist held fixed and the fingers bending towards the camera in synchrony. Running instances of both through our view learning method, with a base threshold of $\theta_0$=0.6 and a "new model" threshold of $\theta_1 = 0.7$, the clustering method found 4 four spatial templates to span all of the images in the both sequences Figure 3 shows the pixel values for these four models.

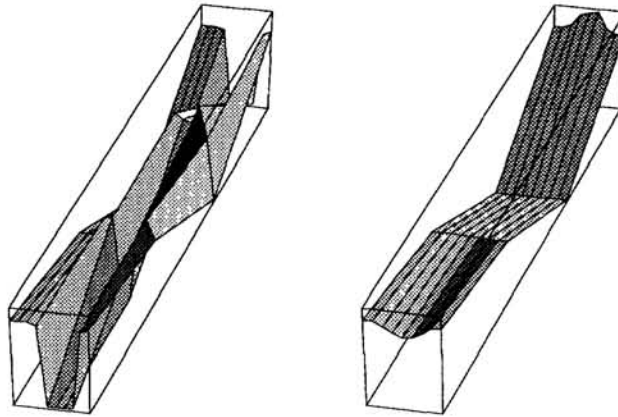

Figure 5: Surface plot of temporal templates found by unsupervised clustering method on sequences of two hand-waving gestures. Vertical axis is score, horizontal axis is time, and depth axis is spatial view index.

## 3.1 TEMPORAL VIEWS

The previous sections provide a method for finding spatial views to reduce the dimensionality in a tracking task. The same method can be applied in the temporal domain as well, using a set of "temporal views". Figure 4 shows an overview of these two stages. We construct temporal views using a similar method to that used for spatial views, but with temporal segmentation cues provided by the user. Sequences of spatial-feature vector outputs (the normalized correlation scores of the spatial views) are passed as input to the unsupervised clustering method, yielding a set of temporal views. To find the distance between two sequences, we again use a normalized correlation metric, with Dynamic Time Warping (DTW) method [1, 10]. This allows the time course of a gesture to vary, as long as the same series of spatial poses is present.

In this way a set of temporal views acting on spatial views which in turn act on image intensities, is created. The responses of these composite views yield a single spatio-temporal stimulus vector which describes spatial *and* temporal properties of the input signal. As an example, for the "hand-waving" example shown above, two temporal views were found by the clustering method. These are shown as surface plots in Figure 5. Empirically we have found that the spatio-temporal units capture the salient aspects of the spatial and temporal variation of the hand gestures in a low-dimensional representation, so efficient classification is possible. The response of these temporal view units on an input sequence containing three instances of each gesture is shown in Figure 6.

## 4    CLASSIFICATION OF GESTURES

The spatio-temporal units obtained by the unsupervised procedure described above are used as inputs to a supervised learning/classification stage (Figure 7(a)). We have implemented two different classification strategies, a traditional Diagonal Gaussian Classifier, and a multi-layer perceptron.

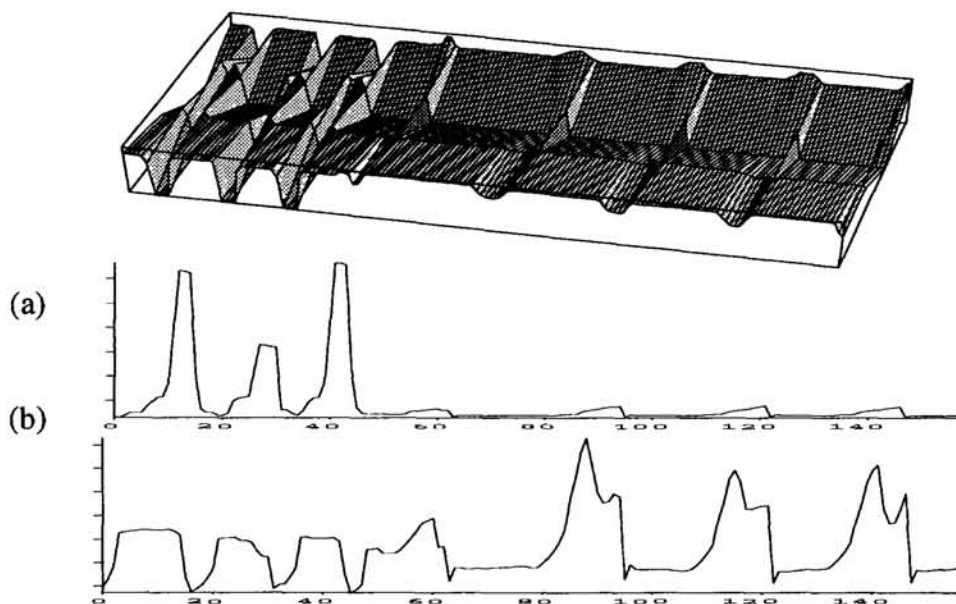

(a)

(b)

Figure 6: (a) surface plot of spatial view responses on input sequence containing three instances of each hand-waving gesture. (b) final spatio-temporal view unit response: the time-warped, normalized correlation score of temporal views on spatial view feature vectors.

As an experiment, we collected 42 examples of a "hello" gesture, 26 examples of "good-bye" and 10 examples of other gestures intended to generate false alarms in the classifier. All gestures were performed by a single user under similar imaging conditions. For each trial we randomly selected half of the target gestures to train the classifier, and tested on the remaining half. (All of the conflictor gestures were used in both training and testing sets since they were few in number.)

Figure 7(b) summarizes the results for the different classification strategies. The Gaussian classifier (DG) achieved an hit rate of 67%, with zero false alarms. The multi-layer perceptron (MLP) was more powerful but less conservative, with a hit rate of 86% and a false alarm rate of 5%. We found the results of the MLP classifier to be quite variable; on many of the trials the classifier was stuck in a local minima and failed to converge on the test set. Additionally there was considerable dependence on the number of units in the hidden layer; empirically we found 12 gave best performance. Nonetheless, the MLP classifier provided good performance. When we excluded the trials on which the classifier failed to converge on the training set, the performance increased to 91% hit rate, 2% false alarm rate.

## 5   CONCLUSION

We have demonstrated a system for tracking and recognition of simple hand gestures. Our entire recognition system, including time-warping and classification, runs in real time (over 10Hz). This is made possible through the use of a special purpose normalized correlation search co-processor. Since the dimensionality of the feature space is low, the dynamic time warping and classifications steps can be implemented on conventional workstations and still achieve real-time performance. Because of this real-time performance, our system is

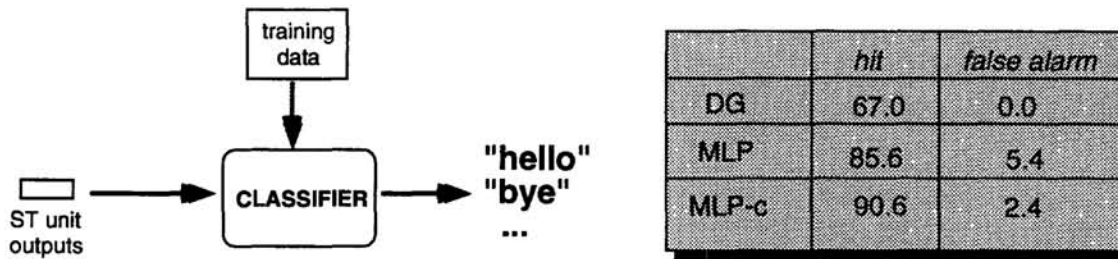

| | hit | false alarm |
|---|---|---|
| DG | 67.0 | 0.0 |
| MLP | 85.6 | 5.4 |
| MLP-c | 90.6 | 2.4 |

Figure 7: Overview of supervised classification stage and results obtained for different types of classifiers.

directly applicable to interactive "glove-free" gestural user interfaces.

## References

[1] Bellman, R. E., (1957) *Dynamic Programming.* Princeton, NJ: Princeton Univ. Press.

[2] Beymer, D., Shashua, A., and Poggio, T., (1993) "Example Based Image Analysis and Synthesis", MIT AI Lab Memo No. 1431

[3] Breuel, T., (1992) "View-based Recognition", *IAPR Workshop on Machine Vision Applications.*

[4] Cipolla, R., Okamotot, Y., and Kuno, Y., (1992) "Qualitative visual interpretation of 3D hand gestures using motion parallax", *IAPR Workshop on Machine Vision Applications.*

[5] Fukumoto, M., Mase, K., and Suenaga, Y., (1992) "Real-Time Detection of Pointing Actions for a Glove-Free Interface", *IAPR Workshop on Machine Vision Applications.*

[6] Ishibuchi, K., Takemura, H., and Kishino, F., "Real-Time Hand Shape Recognition using Pipe-line Image Processor", (1992) *IEEE Workshop on Robot and Human Communication,* pp. 111-116.

[7] Makhoul, J., Roucos, S., and Gish, H., (1985) "Vector Quantization in Speech Coding" *Proc. IEEE,* Vol. 73, No. 11, pp. 1551-1587.

[8] Murase, H.,and Nayar, S. K., (1993) "Learning and Recognition of 3D Objects from Appearance", *Proc. IEEE Qualitative Vision Workshop,* New York City, pp. 39-49.

[9] Poggio, T., and Edelman, S., (1990) "A Network that Learns to Recognize Three Dimensional Objects," *Nature,* Vol. 343, No. 6255, pp. 263-266.

[10] Sakoe, H., and Chiba, S., (1980) "Dynamic Programming optimization for spoken word recognition", *IEEE Trans. ASSP,* Vol. 26, pp. 623-625.

[11] Turk, M., and Pentland, A. P., (1991) "Eigenfaces for Recognition", *Journal of Cognitive Neuroscience,* vol. 3, pp. 71-89.

[12] Ullman, S., and Basri, R., (1991)"Recognition by Linear Combinations of Models," *IEEE PAMI,* Vol. 13, No. 10, pp. 992-1007.
